# Complexity of Finite Precision Neural Network Classifier

**Amir Dembo**[1]
Inform. Systems Lab.
Stanford University
Stanford, Calif. 94305

**Kai-Yeung Siu**
Inform. Systems Lab.
Stanford University
Stanford, Calif. 94305

**Thomas Kailath**
Inform. Systems Lab.
Stanford University
Stanford, Calif. 94305

## ABSTRACT

A rigorous analysis on the finite precision computational aspects of neural network as a pattern classifier via a probabilistic approach is presented. Even though there exist negative results on the capability of perceptron, we show the following positive results: Given $n$ pattern vectors each represented by $cn$ bits where $c > 1$, that are uniformly distributed, with high probability the perceptron can perform all possible binary classifications of the patterns. Moreover, the resulting neural network requires a vanishingly small proportion $O(\log n/n)$ of the memory that would be required for complete storage of the patterns. Further, the perceptron algorithm takes $O(n^2)$ arithmetic operations with high probability, whereas other methods such as linear programming takes $O(n^{3.5})$ in the worst case. We also indicate some mathematical connections with VLSI circuit testing and the theory of random matrices.

## 1    Introduction

It is well known that the perceptron algorithm can be used to find the appropriate parameters in a linear threshold device for pattern classification, provided the pattern vectors are linearly separable. Since the number of parameters in a perceptron is significantly fewer than that needed to store the whole data set, it is tempting to

conclude that when the patterns are linearly separable, the perceptron can achieve a reduction in storage complexity. However, Minsky and Papert [1] have shown an example in which both the learning time and the parameters increase exponentially, when the perceptron would need much more storage than does the whole list of patterns.

Ways around such examples can be explored by noting that analysis that assumes real arithmetic and disregards finite precision aspects might yield misleading results. For example, we present below a simple network with one real valued weight that can simulate all possible classifications of $n$ real valued patterns into $k$ classes, when unlimited accuracy and continuous distribution of the patterns are assumed. For simplicity, let us assume the patterns are real numbers in $[0, 1]$. Consider the following sequence $\{x_{i,j}\}$ generated by each pattern $x_i$ for $i = 1, \ldots, n$:

$$x_{i,1} = k \cdot x_i \mod k$$
$$x_{i,j} = k \cdot x_{i,j-1} \mod k \quad for \ j > 1$$
$$\sigma(x_i, j) = [x_{i,j}]$$

where $[\,]$ denotes the integer part.

Let $f : \{x_1, \ldots, x_n\} \to \{0, \ldots, k-1\}$ denote the desired classification of the patterns. It is easy to see that for any continuous distribution on $[0, 1]$, there exists a $j$ such that $\sigma(x_i, j) = f(x_i)$, with probability one. So, the network $y = \sigma(x, w)$ may simulate any classification with $w = j$ determined from the desired classification as shown above.

So in this paper, we emphasize the finite precision computational aspects of pattern classification problems and provide partial answers to the following questions:

- *Can the perceptron be used as an efficient form of memory?*

- *Does the 'learning' time of perceptron become too long to be practical most of the time even when the patterns are assumed to be linearly separable?*

- *How do the convergence results compare to those obtained by solving system of linear inequalities?*

We attempt to answer the above questions by using a probabilistic approach. The theorems will be presented without proofs; details of the proof will appear in a complete paper. In the following analysis, the phrase 'with high probability' means the probability of the underlying event goes to 1 as the number of patterns goes to

infinity. First, we shall introduce the classical model of a perceptron in more details and give some known results on its limitation as a pattern classifier.

## 2    The Perceptron

A perceptron is a linear threshold device which computes a linear combination of the coordinates of the pattern vector, compares the value with a threshold and outputs $+1$ or $-1$ if the value is larger or smaller than the threshold respectively. More formally, we have
Output:

$$\text{sign}\{<\vec{w}, \vec{x}> -\theta\} = \text{sign}\{\sum_{i=1}^{d} x_i \cdot w_i - \theta\}$$

Input:

$$\vec{x} = (x_1, \ldots, x_d) \in R^d$$

Parameters:

$$\text{weights} \quad \vec{w} = (w_1, \ldots, w_d) \in R^d$$

$$\text{threshold} \quad \theta \in R$$

$$\text{sign}\{y\} = \left\{ \begin{array}{ll} +1 & \text{if } y \geq 0 \\ -1 & \text{otherwise} \end{array} \right.$$

Given m patterns $\vec{x_1}, \ldots, \vec{x_m}$ in $R^d$, there are $2^m$ possible ways of classifying each of the patterns to $\pm 1$. When a desired classification of the patterns is achieveable by a perceptron, the patterns are said to be linearly separable. Rosenblatt(1962) [2] showed that if the patterns are linearly separable, then there is a 'learning' algorithm which he called *perceptron learning algorithm* to find the appropriate parameters $\vec{w}$ and $\theta$. Let $\sigma_i = \pm 1$ be the desired classification of the pattern $\vec{x_i}$. Also, let $\vec{y_i} = \sigma_i \cdot \vec{x_i}$. The perceptron learning algorithm runs as follows:

1. Set $k = 1$, choose an initial value of $\vec{w}(k) \neq 0$.
2. Select an $i \in \{1, \ldots, n\}$, set $\vec{y}(k) = \vec{y_i}$.
3. If $\vec{w}(k) \cdot \vec{y}(k) \geq 0$, goto 2. Else
4. Set $\vec{w}(k+1) = \vec{w}(k) + \vec{y}(k)$, $k = k+1$, goto 2.

The algorithm terminates when step 3 is true for all $\vec{y_i}$. If the patterns are linearly separable, then the above perceptron algorithm is guaranteed to converge in finitely many iterations, i.e. Step 4 would be reached only finitely often.

The existence of such simple and elegant 'learning' algorithm had brought a great deal of interests during the 60's. However, the capability of the perceptron is very limited since only a small portion of the $2^m$ possible binary classifications can be achieved. In fact, Cover(1965) [3] has shown that a perceptron can at most classify the patterns into

$$2\sum_{i=0}^{d-1} \binom{m-1}{i} = O(m^{d-1})$$

different ways out of the $2^m$ possibilities.

The above upper bound $O(m^{d-1})$ is achieved when the pattern vectors are *in general position* i.e. every subset of $d$ vectors in $\{\vec{x_1}, \ldots, \vec{x_m}\}$ are linearly independent. An immediate generalization of this result is the following:

**Theorem 1** *For any function $f(\vec{w}, \vec{x})$ which lies in a function space of dimension $r$, i.e. if we can write*

$$f(\vec{w}, \vec{x}) = \alpha_1(\vec{w})f_1(\vec{x}) + \ldots + \alpha_r(\vec{w})f_r(\vec{x})$$

*then the number of possible classifications of $m$ patterns by sign$\{f(\vec{w}, \vec{x})\}$ is bounded by $O(m^{r-1})$*

## 3  A New Look at the Perceptron

The reason why perceptron is so limited in its capability as a pattern classifier is that the dimension of the pattern vector space is kept fixed while the number of patterns is increased. We consider the binary expansion of each coordinate and view the real pattern vector as a binary vector, but in a much higher dimensional space. The intuition behind this is that we are now making use of every bit of information in the pattern. Let us assume that each pattern vector has dimension $d$ and that each coordinate is given with $m$ bits of accuracy, which grows with the number of patterns $n$ in such a way that $d \cdot m = c \cdot n$ for some $c > 1$. By considering the binary expansion, we can treat the patterns as binary vectors, i.e. each vector belongs to $\{+1, -1\}^{cn}$. If we want to classify the patterns into $k$ classes, we can use $\log k$ number of binary classifiers, each classifying the patterns into the corresponding bit of the binary encoding of the k classes. So without loss of generality, we assume that the number of classes equals 2. Now the classification problem can be viewed as an implementation of a partial Boolean function whose value is only specified on

$n$ inputs out of the $2^{cn}$ possible ones. For arbitrary input patterns, there does not seem to exist an efficient way other than complete storage of the patterns and the use of a look-up table for classification, which will require $O(n^2)$ bits. It is natural to ask if this is the best we can do. Surprisingly, using probabilistic method in combinatorics [4] (counting arguments), we can show the following:

**Theorem 2** *For $n$ sufficiently large, there exists a system that can simulate all possible binary classifications with parameter storage of $n + 2\log n$ bits.*

Moreover, a recent result from the theory of VLSI testing [5], implies that at least $n + \log n$ bits are needed. As the proof of theorem 1 is non-constructive, both the learning of the parameters and the retrieval of the desired classification in the 'optimal' system may be too complex for any practical purpose. Besides, since there is almost no redundancy in the storage of parameters in such an 'optimal' system, there will be no 'generalization' properties. i.e. It is difficult to predict what the output of the system would be on patterns that are not trained. However, a perceptron classifier, while sub-optimal in terms of Theorem 3 below, requires only $O(n \log n)$ bits for parameter storage, compared with $O(n^2)$ bits for a table look up classifier. In addition, it will exhibit 'generalization' properties in the sense that new patterns that are close in Hamming distance to those trained patterns are likely to be classified into the same class. So, if we allow some vanishingly small probability of error, we can give an affirmative answer to the first question raised at the beginning:

**Theorem 3** *Assume the $n$ pattern vectors are uniformly distributed over $\{+1, -1\}^{cn}$, then with high probability, the patterns can be classified into all $2^n$ possible ways using perceptron algorithm. Further, the storage of parameters requires only $O(n \log n)$ bits.*

In other words, when the input patterns are given with high precision, perceptron can be used as an efficient form of memory.

The known upper bound on the learning time of perceptron depends on the maximum length of the input pattern vectors, and the minimum distance $\delta$ of the pattern vectors to a separating hyperplane. In the following analysis, our probabilistic assumption guarantees the pattern vectors to be linearly independent with high probability and thus linearly separable. In order to give an probabilistic upper bound on the learning time of the perceptron, we first give a lower bound on the minimum distance $\delta$ with high probability:

**Lemma 1** *Let $n$ be the number of pattern vectors each in $R^m$, where $m = (1 + \epsilon)n$ and $\epsilon$ is any constant $> 0$. Assume the entries of each vector $v$ are iid random variables with zero mean and bounded second moment. Then with probability $\rightarrow 1$*

*as $n \to \infty$ , there exists a separating hyperplane and a $\delta^* > 0$ such that each vector is at a distance of at least $\delta^*$ from it.*

In our case, each coordinate of the patterns is assumed to be equally likely $\pm 1$ and clearly the conditions in the above lemma are satisfied. In general, when the dimension of the pattern vectors is larger than and increases linearly with the number of patterns, the above theorem applies provided the patterns are given with high enough precision that a continuous distribution is a sufficiently good model for analysis.

The above lemma makes use of a famous conjecture from the theory of random matrices [6] which gives a lower bound on the minimum singular value of a random matrix. We actually proved the conjecture during our course of study, which states which states that the minimum singular value of a $cn$ by $n$ random matrix with $c > 1$, grows as $\sqrt{n}$ almost surely.

**Theorem 4** *Let $A_n$ be a $cn \times n$ random matrix with $c > 1$, whose entries are i.i.d. entries with zero mean and bounded second moment, $\sigma(\cdot)$ denote the minimum singular value of a matrix. Then there exists $\beta > 0$ such that*

$$\liminf_{n \to \infty} \sigma(\frac{A_n}{\sqrt{n}}) > \beta$$

*with probability 1.*

Note that our probabilistic assumption on the patterns includes a wide class of distributions, in particular the zero mean normal and symmetric uniform distribution on a bounded interval. In addition, they satisfy the following condition:

(*) There exists a $\alpha > 0$ such that $P\{|v| > \alpha \sqrt{n}\} \to 0$ as $n \to \infty$.

Before we answer the last two questions raised at the beginning, we state the following known result on the perceptron algorithm as a second lemma:

**Lemma 2** *Suppose there exists a unit vector $w^*$ such that $w^* \cdot v > \delta$ for some $\delta > 0$ and for all pattern vectors $v$. Then the perceptron algorithm will converge to a solution vector in $\leq N^2/\delta^2$ number of iterations, where $N$ is the maximum length of the pattern vectors.*

Now we are ready to state the following

**Theorem 5** *Suppose the patterns satisfy the probabilistic assumptions stated in*

*Lemma 1 and the condition* (∗), *then with high probability, the perceptron takes* $O(n^2)$ *arithmetic operations to terminate.*

As mentioned earlier, another way of finding a separating hyperplane is to solve a system of linear inequalities using linear programming, which requires $O(n^{3.5})$ arithmetic operations [7]. Under our probabilistic assumptions, the patterns are linearly independent with high probability, so that we can actually solve a system of linear equations. However, this still requires $O(n^3)$ arithmetic operations. Further, these methods require batch processing in the sense that all patterns have to be stored in advance in order to find the desired parameters, in constrast to the sequential 'learning' nature of the perceptron algorithm. So for training this neural network classifier, perceptron algorithm seems more preferable.

When the number of patterns is polynomial in the total number of bits representing each pattern, we may first extend each vector to a dimension at least as large as the number of patterns, and then apply the perceptron to compress the storage of parameters. One way of adding these extra bits is to form products of the coordinates within each pattern. Note that by doing so, the coordinates of each pattern are pairwise independent. We conjecture that theorem 3 still applies, implying even more reduction in storage requirements. Simulation results strongly support our conjecture.

## 4    Conclusion

In this paper, the finite precision computational aspects of pattern classification problems are emphasized. We show that the perceptron, in contrast to common belief, can be quite efficient as a pattern classifier, provided the patterns are given with high enough precision. Using a probabilistic approach, we show that the perceptron algorithm can even outperform linear programming under certain conditions. During the course of this work, we also discovered some mathematical connections with VLSI circuit testing and the theory of random matrices. In particular, we have proved an open conjecture regarding the minimum singular value of a random matrix.

## Acknowledgements

This work was supported in part by the Joint Services Program at Stanford University (US Army, US Navy, US Air Force) under Contract DAAL03-88-C-0011, and NASA Headquarters, Center for Aeronautics and Space Information Sciences (CASIS) under Grant NAGW-419-S5.

## Footnotes

[1] The coauthor is now with the Mathematics and Statistics Department of Stanford University.

# References

[1] M. Minsky and S. Papert, *Perceptrons*, The MIT Press, expanded edition, 1988.

[2] F. Rosenblatt, *Principles of Neurodynamics*, Spartan Books, New York, 1962.

[3] T. M. Cover, "Geometrical and Statistical Properties of Systems of Linear Inequalities with Applications in Pattern Recognition", *IEEE Trans. on Electronic Computers*, **EC-14**:326–34, 1965.

[4] P. Erdös and J. Spencer, *Probabilistic Methods in Combinatorics*, Academic Press/Akademiai Kiado, New York-Budapest, 1974.

[5] G. Seroussi and N. Bshouty, "Vector Sets for Exhaustive Testing of Logic Circuits", *IEEE Trans. Inform. Theory*, **IT-34**:513–522, 1988.

[6] J. Cohen, H. Kesten and C. Newman, editor, *Random Matrices and Their Applications*, volume 50 of *Contemporary Mathematics*, American Mathematical Society, 1986.

[7] N. Karmarkar, "A New Polynomial-Time Algorithm for Linear Programming", *Combinatorica 1*, pages 373–395, 1984.
